# Learning Nonlinear Dynamical Systems using an EM Algorithm

**Zoubin Ghahramani and Sam T. Roweis**

Gatsby Computational Neuroscience Unit
University College London
London WC1N 3AR, U.K.

http://www.gatsby.ucl.ac.uk/

## Abstract

The Expectation–Maximization (EM) algorithm is an iterative procedure for maximum likelihood parameter estimation from data sets with missing or hidden variables [2]. It has been applied to system identification in linear stochastic state-space models, where the state variables are hidden from the observer and both the state and the parameters of the model have to be estimated simultaneously [9]. We present a generalization of the EM algorithm for parameter estimation in *nonlinear* dynamical systems. The "expectation" step makes use of Extended Kalman Smoothing to estimate the state, while the "maximization" step re-estimates the parameters using these uncertain state estimates. In general, the nonlinear maximization step is difficult because it requires integrating out the uncertainty in the states. However, if Gaussian radial basis function (RBF) approximators are used to model the nonlinearities, the integrals become tractable and the maximization step can be solved via systems of linear equations.

## 1   Stochastic Nonlinear Dynamical Systems

We examine inference and learning in discrete-time dynamical systems with hidden state $x_t$, inputs $u_t$, and outputs $y_t$.[1] The state evolves according to stationary nonlinear dynamics driven by the inputs and by additive noise

$$x_{t+1} = f(x_t, u_t) + w \tag{1}$$

where $w$ is zero-mean Gaussian noise with covariance $Q$. [2] The outputs are non-linearly related to the states and inputs by

$$y_t = g(x_t, u_t) + v \tag{2}$$

where $v$ is zero-mean Gaussian noise with covariance $R$. The vector-valued nonlinearities $f$ and $g$ are assumed to be differentiable, but otherwise arbitrary.

Models of this kind have been examined for decades in various communities. Most notably, nonlinear state-space models form one of the cornerstones of modern systems and control engineering. In this paper, we examine these models within the framework of probabilistic graphical models and derive a novel learning algorithm for them based on EM. With one exception,[3] this is to the best of our knowledge the first paper addressing learning of stochastic nonlinear dynamical systems of the kind we have described within the framework of the EM algorithm.

The classical approach to system identification treats the parameters as hidden variables, and applies the Extended Kalman Filtering algorithm (described in section 2) to the nonlinear system with the state vector augmented by the parameters [5].[4] This approach is inherently on-line, which may be important in certain applications. Furthermore, it provides an estimate of the covariance of the parameters at each time step. In contrast, the EM algorithm we present is a batch algorithm and does not attempt to estimate the covariance of the parameters.

There are three important advantages the EM algorithm has over the classical approach. First, the EM algorithm provides a straightforward and principled method for handing missing inputs or outputs. Second, EM generalizes readily to more complex models with combinations of discrete and real-valued hidden variables. For example, one can formulate EM for a *mixture* of nonlinear dynamical systems. Third, whereas it is often very difficult to prove or analyze stability within the classical on-line approach, the EM algorithm is always attempting to maximize the likelihood, which acts as a Lyapunov function for stable learning.

In the next sections we will describe the basic components of the learning algorithm. For the expectation step of the algorithm, we infer the conditional distribution of the hidden states using Extended Kalman Smoothing (section 2). For the maximization step we first discuss the general case (section 3) and then describe the particular case where the nonlinearities are represented using Gaussian radial basis function (RBF; [6]) networks (section 4).

## 2   Extended Kalman Smoothing

Given a system described by equations (1) and (2), we need to infer the hidden states from a history of observed inputs and outputs. The quantity at the heart of this inference problem is the conditional density $P(x_t | u_1, \ldots, u_T, y_1, \ldots, y_T)$, for $1 \leq t \leq T$, which captures the fact that the system is stochastic and therefore our inferences about $x$ will be uncertain.

For linear dynamical systems with Gaussian state evolution and observation noises, this conditional density is Gaussian and the recursive algorithm for computing its mean and covariance is known as *Kalman smoothing* [4, 8]. Kalman smoothing is directly analogous to the forward–backward algorithm for computing the conditional hidden state distribution in a hidden Markov model, and is also a special case of the belief propagation algorithm.[5]

For nonlinear systems this conditional density is in general non-Gaussian and can in fact be quite complex. Multiple approaches exist for inferring the hidden state distribution of such nonlinear systems, including sampling methods [7] and variational approximations [3]. We focus instead in this paper on a classic approach from engineering, *Extended Kalman Smoothing* (EKS).

Extended Kalman Smoothing simply applies Kalman smoothing to a local *linearization* of the nonlinear system. At every point $\tilde{x}$ in $x$-space, the derivatives of the vector-valued functions $f$ and $g$ define the matrices, $A_{\tilde{x}} \equiv \left. \frac{\partial f}{\partial x} \right|_{x=\tilde{x}}$ and $C_{\tilde{x}} \equiv \left. \frac{\partial g}{\partial x} \right|_{x=\tilde{x}}$, respectively. The dynamics are linearized about $\hat{x}_t$, the mean of the Kalman filter state estimate at time $t$:

$$x_{t+1} = f(\hat{x}_t, u_t) + A_{\hat{x}_t} (x_t - \hat{x}_t) + w. \tag{3}$$

The output equation (2) can be similarly linearized. If the prior distribution of the hidden state at $t = 1$ was Gaussian, then, in this linearized system, the conditional distribution of the hidden state at any time $t$ given the history of inputs and outputs will also be Gaussian. Thus, Kalman smoothing can be used on the linearized system to infer this conditional distribution (see figure 1, left panel).

## 3   Learning

The M step of the EM algorithm re-estimates the parameters given the observed inputs, outputs, and the conditional distributions over the hidden states. For the model we have described, the parameters define the nonlinearities $f$ and $g$, and the noise covariances $Q$ and $R$.

Two complications arise in the M step. First, it may not be computationally feasible to fully re-estimate $f$ and $g$. For example, if they are represented by neural network regressors, a single full M step would be a lengthy training procedure using backpropagation, conjugate gradients, or some other optimization method. Alternatively, one could use partial M steps, for example, each consisting of one or a few gradient steps.

The second complication is that $f$ and $g$ have to be trained using the uncertain state estimates output by the EKS algorithm. Consider fitting $f$, which takes as inputs $x_t$ and $u_t$ and outputs $x_{t+1}$. For each $t$, the conditional density estimated by EKS is a full-covariance Gaussian in $(x_t, x_{t+1})$-space. So $f$ has to be fit not to a set of data points but instead to a mixture of full-covariance Gaussians in input-output space (Gaussian "clouds" of data). Integrating over this type of noise is non-trivial for almost any form of $f$. One simple but inefficient approach to bypass this problem is to draw a large sample from these Gaussian clouds of uncertain data and then fit $f$ to these samples in the usual way. A similar situation occurs with $g$.

In the next section we show how, by choosing Gaussian radial basis functions to model $f$ and $g$, both of these complications vanish.

## 4    Fitting Radial Basis Functions to Gaussian Clouds

We will present a general formulation of an RBF network from which it should be clear how to fit special forms for $f$ and $g$. Consider the following nonlinear mapping from input vectors $x$ and $u$ to an output vector $z$:

$$z = \sum_{i=1}^{I} h_i \, \rho_i(x) + Ax + Bu + b + w, \tag{4}$$

where $w$ is a zero-mean Gaussian noise variable with covariance $Q$. For example, one form of $f$ can be represented using (4) with the substitutions $x \leftarrow x_t$, $u \leftarrow u_t$, and $z \leftarrow x_{t+1}$; another with $x \leftarrow (x_t, u_t)$, $u \leftarrow \emptyset$, and $z \leftarrow x_{t+1}$. The parameters are: the coefficients of the $I$ RBFs, $h_i$; the matrices $A$ and $B$ multiplying inputs $x$ and $u$, respectively; and an output bias vector $b$. Each RBF is assumed to be a Gaussian in $x$-space, with center $c_i$ and width given by the covariance matrix $S_i$:

$$\rho_i(x) = |2\pi S_i|^{-1/2} \exp\left\{ -\frac{1}{2}(x - c_i)^{\mathsf{T}} S_i^{-1}(x - c_i) \right\}. \tag{5}$$

The goal is to fit this model to data $(u, x, z)$. The complication is that the data set comes in the form of a mixture of Gaussian distributions. Here we show how to analytically integrate over this mixture distribution to fit the RBF model.

Assume the data set is:

$$P(x, z, u) = \frac{1}{J} \sum_j \mathcal{N}_j(x, z) \, \delta(u - u_j). \tag{6}$$

That is, we observe samples from the $u$ variables, each paired with a Gaussian "cloud" of data, $\mathcal{N}_j$, over $(x, z)$. The Gaussian $\mathcal{N}_j$ has mean $\mu_j$ and covariance matrix $C_j$.

Let $\hat{z}_\theta(x, u) = \sum_{i=1}^{I} h_i \, \rho_i(x) + Ax + Bu + b$, where $\theta$ is the set of parameters $\theta = \{h_1 \ldots h_I, A, B, b\}$. The log likelihood of a single data point under the model is:

$$-\frac{1}{2}\left[z - \hat{z}_\theta(x, u)\right]^{\mathsf{T}} Q^{-1} \left[z - \hat{z}_\theta(x, u)\right] - \frac{1}{2}\ln|Q| + \text{const.}$$

The maximum likelihood RBF fit to the mixture of Gaussian data is obtained by minimizing the following integrated quadratic form:

$$\min_{\theta, Q} \left\{ \sum_j \int_x \int_z \mathcal{N}_j(x, z) \left[z - \hat{z}_\theta(x, u_j)\right]^{\mathsf{T}} Q^{-1} \left[z - \hat{z}_\theta(x, u_j)\right] dx \, dz + J\ln|Q| \right\}. \tag{7}$$

We rewrite this in a slightly different notation, using angled brackets $\langle \cdot \rangle_j$ to denote expectation over $\mathcal{N}_j$, and defining

$$\begin{aligned} \theta &\equiv [h_1^{\mathsf{T}} \, h_2^{\mathsf{T}} \ldots h_I^{\mathsf{T}} \, A^{\mathsf{T}} \, B^{\mathsf{T}} \, b^{\mathsf{T}}]^{\mathsf{T}} \\ \Phi &\equiv [\rho_1(x) \, \rho_2(x) \ldots \rho_I(x) \, x \, u \, 1]. \end{aligned}$$

Then, the objective can be written

$$\min_{\theta, Q} \left\{ \sum_j \left\langle (z - \theta \, \Phi)^{\mathsf{T}} Q^{-1}(z - \theta \, \Phi) \right\rangle_j + J\ln|Q| \right\}. \tag{8}$$

Taking derivatives with respect to $\theta$, premultiplying by $-Q^{-1}$, and setting to zero gives the linear equations $\sum_j \langle (z - \theta\Phi)\Phi^\top \rangle_j = 0$, which we can solve for $\theta$ and $Q$:

$$\hat\theta = \left( \sum_j \langle z\Phi^\top \rangle_j \right) \left( \sum_j \langle \Phi\Phi^\top \rangle_j \right)^{-1}, \quad \hat Q = \frac{1}{J}\left( \sum_j \langle zz^\top \rangle_j - \hat\theta \sum_j \langle \Phi z^\top \rangle_j \right). \quad (9)$$

In other words, given the expectations in the angled brackets, the optimal parameters can be solved for via a set of linear equations. In appendix A we show that these expectations can be computed analytically. The derivation is somewhat laborious, but the intuition is very simple: the Gaussian RBFs multiply with the Gaussian densities $\mathcal{N}_j$ to form new unnormalized Gaussians in $(x, y)$-space. Expectations under these new Gaussians are easy to compute. This fitting algorithm is illustrated in the right panel of figure 1.

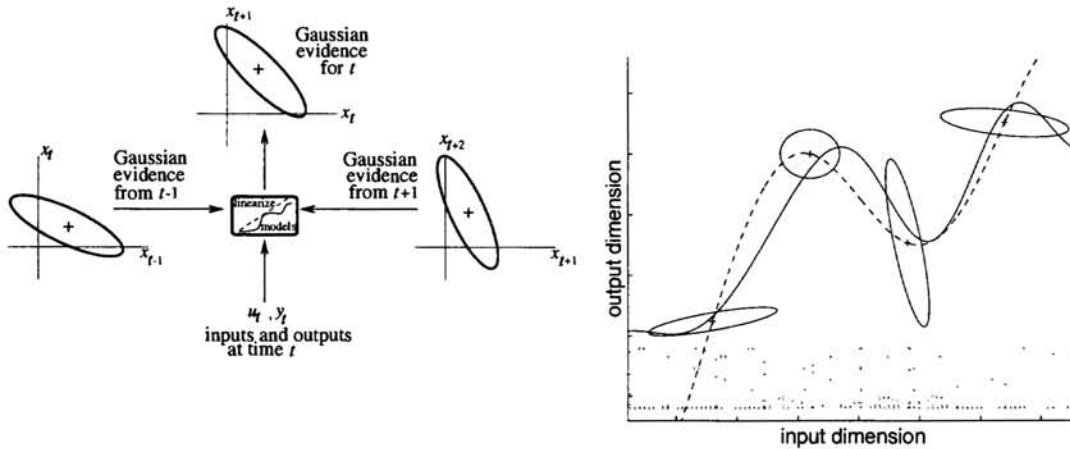

Figure 1: Illustrations of the E and M steps of the algorithm. The left panel shows the information used in Extended Kalman Smoothing (EKS), which infers the hidden state distribution during the E-step. The right panel illustrates the regression technique employed during the M-step. A fit to a mixture of Gaussian densities is required; if Gaussian RBF networks are used then this fit can be solved analytically. The dashed line shows a regular RBF fit to the centres of the four Gaussian densities while the solid line shows the analytic RBF fit using the covariance information. The dotted lines below show the support of the RBF kernels.

## 5   Results

We tested how well our algorithm could learn the dynamics of a nonlinear system by observing only its inputs and outputs. The system consisted of a single input, state and output variable at each time, where the relation of the state from one time step to the next was given by a tanh nonlinearity. Sample outputs of this system in response to white noise are shown in figure 2 (left panel).

We initialized the nonlinear model with a linear dynamical model trained with EM, which in turn we initialized with a variant of factor analysis. The model was given 11 RBFs in $x_t$-space, which were uniformly spaced within a range which was automatically determined from the density of points in $x_t$-space. After the initialization was over, the algorithm discovered the sigmoid nonlinearity in the dynamics within less than 10 iterations of EM (figure 2, middle and right panels).

Further experiments need to be done to determine how practical this method will be in real domains.

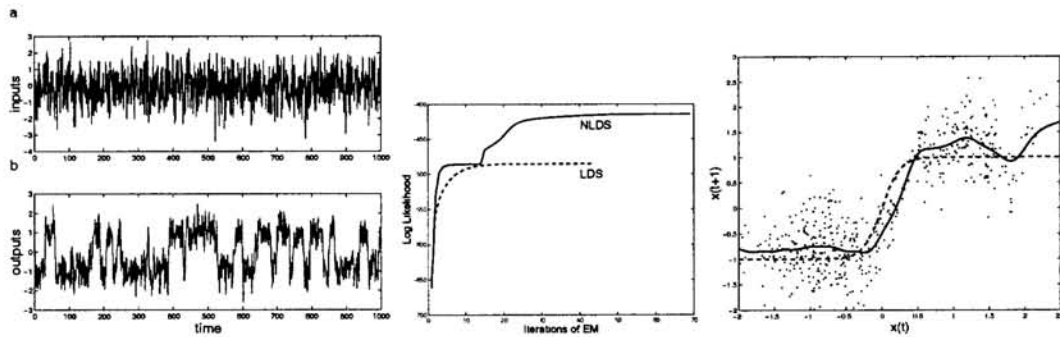

Figure 2: **(left):** Data set used for training (first half) and testing (rest), which consists of a time series of inputs, $u_t$ **(a)**, and outputs $y_t$ **(b)**. **(middle):** Representative plots of log likelihood vs iterations of EM for linear dynamical systems (dashed line) and nonlinear dynamical systems trained as described in this paper (solid line). Note that the actual likelihood for nonlinear dynamical systems cannot generally be computed analytically; what is shown here is the approximate likelihood computed by EKS. The kink in the solid curve comes when initialization with linear dynamics ends and the nonlinearity starts to be learned. **(right):** Means of $(x_t, x_{t+1})$ Gaussian posteriors computed by EKS (dots), along with the sigmoid nonlinearity (dashed line) and the RBF nonlinearity learned by the algorithm. At no point does the algorithm actually observe $(x_t, x_{t+1})$ pairs; these are inferred from inputs, outputs, and the current model parameters.

## 6 Discussion

This paper brings together two classic algorithms, one from statistics and another from systems engineering, to address the learning of stochastic nonlinear dynamical systems. We have shown that by pairing the Extended Kalman Smoothing algorithm for state estimation in the E-step, with a radial basis function learning model that permits analytic solution of the M-step, the EM algorithm is capable of learning a nonlinear dynamical model from data. As a side effect we have derived an algorithm for training a radial basis function network to fit data in the form of a mixture of Gaussians.

Our initial approach has three potential limitations. First, the M-step presented does not modify the centres or widths of the RBF kernels. It is possible to compute the expectations required to change the centres and widths, but it requires resorting to a partial M-step. For low dimensional state spaces, filling the space with pre-fixed kernels is feasible, but this strategy needs exponentially many RBFs in high dimensions. Second, EM training can be slow, especially if initialized poorly. Understanding how different hidden variable models are related can help devise sensible initialization heuristics. For example, for this model we used a nested initialization which first learned a simple linear dynamical system, which in turn was initialized with a variant of factor analysis. Third, the method presented here learns from batches of data and assumes stationary dynamics. We have recently extended it to handle online learning of nonstationary dynamics.

The belief network literature has recently been dominated by two methods for approximate inference, Markov chain Monte Carlo [7] and variational approximations [3]. To our knowledge this paper is the first instance where extended Kalman smoothing has been used to perform approximate inference in the E step of EM. While EKS does not have the theoretical guarantees of variational methods, its simplicity has gained it wide acceptance in the estimation and control literatures as a method for doing inference in nonlinear dynamical systems. We are now exploring generalizations of this method to learning nonlinear multilayer belief networks.

## Acknowledgements

ZG would like to acknowledge the support of the CITO (Ontario) and the Gatsby Charitable Fund. STR was supported in part by the NSF Center for Neuromorphic Systems Engineering and by an NSERC of Canada 1967 Award.

## A  Expectations Required to Fit the RBFs

The expectations we need to compute for equation 9 are $\langle x \rangle_j$, $\langle z \rangle_j$, $\langle xx^\top \rangle_j$, $\langle xz^\top \rangle_j$, $\langle zz^\top \rangle_j$, $\langle \rho_i(x) \rangle_j$, $\langle x\, \rho_i(x) \rangle_j$, $\langle z\, \rho_i(x) \rangle_j$, $\langle \rho_i(x)\, \rho_\ell(x) \rangle_j$.

Starting with some of the easier ones that do not depend on the RBF kernel $\rho$:

$$\begin{aligned}
\langle x \rangle_j &= \mu_j^x & \langle z \rangle_j &= \mu_j^z \\
\langle xx^\top \rangle_j &= \mu_j^x \mu_j^{x,T} + C_j^{xx} & \langle xz^\top \rangle_j &= \mu_j^x \mu_j^{z,T} + C_j^{xz} \\
\langle zz^\top \rangle_j &= \mu_j^z \mu_j^{z,T} + C_j^{zz}
\end{aligned}$$

Observe that when we multiply the Gaussian RBF kernel $\rho_i(x)$ (equation 5) and $\mathcal{N}_j$ we get a Gaussian density over $(x, z)$ with mean and covariance

$$\mu_{ij} = C_{ij}\left(C_j^{-1}\mu_j + \begin{bmatrix} S_i^{-1}c_i \\ 0 \end{bmatrix}\right) \quad \text{and} \quad C_{ij} = \left(C_j^{-1} + \begin{bmatrix} S_i^{-1} & 0 \\ 0 & 0 \end{bmatrix}\right)^{-1},$$

and an extra constant (due to lack of normalization),

$$\beta_{ij} = (2\pi)^{-d_x/2}|S_i|^{-1/2}|C_j|^{-1/2}|C_{ij}|^{1/2}\exp\{-\delta_{ij}/2\}$$

where $\delta_{ij} = c_i^\top S_i^{-1}c_i + \mu_j^\top C_j^{-1}\mu_j - \mu_{ij}^\top C_{ij}^{-1}\mu_{ij}$. Using $\beta_{ij}$ and $\mu_{ij}$, we can evaluate the other expectations:

$$\langle \rho_i(x) \rangle_j = \beta_{ij}, \qquad \langle x\, \rho_i(x) \rangle_j = \beta_{ij}\mu_{ij}^x, \quad \text{and} \quad \langle z\, \rho_i(x) \rangle_j = \beta_{ij}\mu_{ij}^z.$$

Finally, $\langle \rho_i(x)\, \rho_\ell(x) \rangle_j = (2\pi)^{-d_x}|C_j|^{-1/2}|S_i|^{-1/2}|S_\ell|^{-1/2}|C_{i\ell j}|^{1/2}\exp\{-\gamma_{i\ell j}/2\}$, where

$$C_{i\ell j} = \left(C_j^{-1} + \begin{bmatrix} S_i^{-1} + S_\ell^{-1} & 0 \\ 0 & 0 \end{bmatrix}\right)^{-1} \text{and} \quad \mu_{i\ell j} = C_{i\ell j}\left(C_j^{-1}\mu_j + \begin{bmatrix} S_i^{-1}c_i + S_\ell^{-1}c_\ell \\ 0 \end{bmatrix}\right),$$

and $\gamma_{i\ell j} = c_i^\top S_i^{-1}c_i + c_\ell^\top S_\ell^{-1}c_\ell + \mu_j^\top C_j^{-1}\mu_j - \mu_{i\ell j}^\top C_{i\ell j}^{-1}\mu_{i\ell j}$.

## Footnotes

[1] All lowercase characters (except indices) denote vectors. Matrices are represented by uppercase characters.

[2] The Gaussian noise assumption is less restrictive for nonlinear systems than for linear systems since the nonlinearity can be used to generate non-Gaussian state noise.

[3] The authors have just become aware that Briegel and Tresp (this volume) have applied EM to essentially the same model. Briegel and Tresp's method uses multilayer perceptrons (MLP) to approximate the nonlinearities, and requires sampling from the hidden states to fit the MLP. We use Gaussian radial basis functions (RBFs) to model the nonlinearities, which can be fit analytically without sampling (see section 4).

[4] It is important not to confuse this use of the Extended Kalman algorithm, to simultaneously estimate parameters and hidden states, with our use of EKS, to estimate just the hidden state as part of the E step of EM.

[5]The forward part of the Kalman smoother is the Kalman filter.

## References

[1] T. Briegel and V. Tresp. Fisher Scoring and a Mixture of Modes Approach for Approximate Inference and Learning in Nonlinear State Space Models. In *This Volume*. MIT Press, 1999.

[2] A.P. Dempster, N.M. Laird, and D.B. Rubin. Maximum likelihood from incomplete data via the EM algorithm. *J. Royal Statistical Society Series B*, 39:1–38, 1977.

[3] M. I. Jordan, Z. Ghahramani, T. S. Jaakkola, and L. K. Saul. An Introduction to variational methods in graphical models. *Machine Learning*, 1999.

[4] R. E. Kalman and R. S. Bucy. New results in linear filtering and prediction. *Journal of Basic Engineering (ASME)*, 83D:95–108, 1961.

[5] L. Ljung and T. Söderström. *Theory and Practice of Recursive Identification*. MIT Press, Cambridge, MA, 1983.

[6] J. Moody and C. Darken. Fast learning in networks of locally-tuned processing units. *Neural Computation*, 1(2):281–294, 1989.

[7] R. M. Neal. Probabilistic inference using Markov chain monte carlo methods. Technical Report CRG-TR-93-1, 1993.

[8] H. E. Rauch. Solutions to the linear smoothing problem. *IEEE Transactions on Automatic Control*, 8:371–372, 1963.

[9] R. H. Shumway and D. S. Stoffer. An approach to time series smoothing and forecasting using the EM algorithm. *J. Time Series Analysis*, 3(4):253–264, 1982.
